# Rapid Deformable Object Detection using Dual-Tree Branch-and-Bound

**Iasonas Kokkinos**
Center for Visual Computing
Ecole Centrale de Paris
`iasonas.kokkinos@ecp.fr`

## Abstract

In this work we use Branch-and-Bound (BB) to efficiently detect objects with deformable part models. Instead of evaluating the classifier score exhaustively over image locations and scales, we use BB to focus on promising image locations. The core problem is to compute bounds that accommodate part deformations; for this we adapt the Dual Trees data structure [7] to our problem.

We evaluate our approach using Mixture-of-Deformable Part Models [4]. We obtain exactly the same results but are 10-20 times faster on average. We also develop a multiple-object detection variation of the system, where hypotheses for 20 categories are inserted in a common priority queue. For the problem of finding the strongest category in an image this results in a 100-fold speedup.

## 1 Introduction

Deformable Part Models (DPMs) deliver state-of-the-art object detection results [4] on challenging benchmarks when trained discriminatively, and have become a standard in object recognition research. At the heart of these models lies the optimization of a merit function -the classifier score- with respect to the part displacements and the global object pose. In this work we take the classifier for granted, using the models of [4], and focus on the optimization problem.

The most common detection algorithm used in conjunction with DPMs relies on Generalized Distance Transforms (GDTs) [5], whose complexity is linear in the image size. Despite its amazing efficiency this algorithm still needs to first evaluate the score everywhere before picking its maxima.

In this work we use Branch-and-Bound in conjunction with part-based models. For this we exploit the Dual Tree (DT) data structure [7], developed originally to accelerate operations related to Kernel Density Estimation (KDE). We use DTs to provide the bounds required by Branch-and-Bound.

Our method is fairly generic; it applies to any star-shape graphical model involving continuous variables, and pairwise potentials expressed as separable, decreasing binary potential kernels. We evaluate our technique using the mixture-of-deformable part models of [4]. Our algorithm delivers *exactly the same* results, but is 15-30 times faster. We also develop a multiple-object detection variation of the system, where all object hypotheses are inserted in the same priority queue. If our task is to find the best (or k-best) object hypotheses in an image this can result in a 100-fold speedup.

## 2 Previous Work on Efficient Detection

Cascaded object detection [20] has led to a proliferation of vision applications, but far less work exists to deal with part-based models. The combinatorics of matching have been extensively studied for rigid objects [8], while [17] used $A^*$ for detecting object instances. For categories, recent works [1, 10, 11, 19, 6, 18, 15] have focused on reducing the high-dimensional pose search space during

detection by initially simplifying the cost function being optimized, mostly using ideas similar to $A^*$ and coarse-to-fine processing. In the recent work of [4] thresholds pre-computed on the training set are used to prune computation and result in substantial speedups compared to GDTs.

Branch-and-bound (BB) prioritizes the search of promising image areas, as indicated by an upper bound on the classifier's score. A most influential paper has been the Efficient Subwindow Search (ESS) technique of [12], where an upper bound of a bag-of-words classifier score delivers the bounds required by BB. Later [16] combined Graph-Cuts with BB for object segmentation, while in [13] a general cascade system was devised for efficient detection with a nonlinear classifier.

Our work is positioned with respect to these works as follows: unlike existing BB works [16, 12, 15], we use the DPM cost and thereby accommodate parts in a rigorous energy minimization framework. And unlike the pruning-based works [1, 6, 4, 18], we do not make any approximations or assumptions about when it is legitimate to stop computation; our method is exact.

We obtain the bound required by BB from Dual Trees. To the best of our knowledge, Dual Trees have been minimally been used in object detection; we are only aware of the work in [9] which used DTs to efficiently generate particles for Nonparametric Belief Propagation. Here we show that DTs can be used for part-based detection, which is related conceptually, but entirely different technically.

## 3 Preliminaries

We first describe the cost function used in DPMs, then outline the limitations of GDT-based detection, and finally present the concepts of Dual Trees relevant to our setting. Due to lack of space we refer to [2, 4] for further details on DPMs and to [7] [14] for Dual Trees.

### 3.1 Merit function for DPMs

We consider a star-shaped graphical model consisting of a set of $P + 1$ nodes $\{n_0, \ldots n_P\}$; $n_0$ is called the root and the part nodes $n_1, \ldots, n_P$ are connected to the root. Each node $p$ has a unary observation potential $U_p(x)$, indicating the fidelity of the image at $x$ to the node; e.g. in [2] $U_p(x)$ is the inner product of a HOG feature at $x$ with a discriminant $w_p$ for $p$.

The location $x_p = (h_p, v_p)$ of part $p$ is constrained with respect to the root location $x_0 = (h_0, v_0)$ in terms of a quadratic binary potential $B_p(x_p, x_0)$ of the form:

$$B_p(x_p, x_0) = -(x_p - x_0 - \mu_p)^T I_p (x_p - x_0 - \mu_p) = -(h_p - h_0 - \eta_p)^2 H_p - (v_p - v_0 - \nu_p)^2 V_p,$$

where $I_p = \mathrm{diag}(H_p, V_p)$ is a diagonal precision matrix and $m_p = (\eta_p, \nu_p)$ is the nominal difference of root-part locations. We will freely alternate between the vector $x$ and its horizontal/vertical $h/v$ coordinates. Moreover we consider $\eta_0 = 0, \mu_0 = 0$ and $H_0, V_0$ large enough so that $B_0(x_p, x_0)$ will be zero for $x_p = x_0$ and practically infinite elsewhere.

If the root is at $x_0$ the merit for part $p$ being at $x_p$ is given by $m_p(x_p, x_0) = U_p(x_p) + B_p(x_p, x_0)$; summing over $p$ gives the score $\sum_p m_p(x_p, x_0)$ of a root-and-parts configuration $X = (x_0, \ldots, x_P)$. The detector score at point $x$ is obtained by maximizing over those $X$ with $x_0 = x$; this amounts to computing:

$$S(x) \doteq \sum_{p=0}^{P} \max_{x_p} m_p(x_p, x) = \sum_{p=0}^{P} \max_{x_p} U_p(x_p) - (h_p - h - \eta_p)^2 H_p - (v_p - v - \nu_p)^2 V_p. \quad (1)$$

A GDT can be used to maximize each summand in Eq. 1 jointly for all values of $x_0$ in time $O(N)$, where $N$ is the number of possible locations. This is dramatically faster than the naive $O(N^2)$ computation. For a P-part model, complexity decreases from $O(N^2 P)$ to $O(NP)$.

Still, the $N$ factor can make things slow for large images. If we know that a certain threshold will be used for detection, e.g. $-1$ for a classifier trained with SVMs, the GDT-based approach turns out to be wasteful as it treats equally all image locations, even those where we can quickly realize that the classifier score cannot exceed this threshold.

This is illustrated in Fig. 1: in (a) we show the part-root configuration that gives the maximum score, and in (b) the score of a bicycle model from [4] over the whole image domain. Our approach

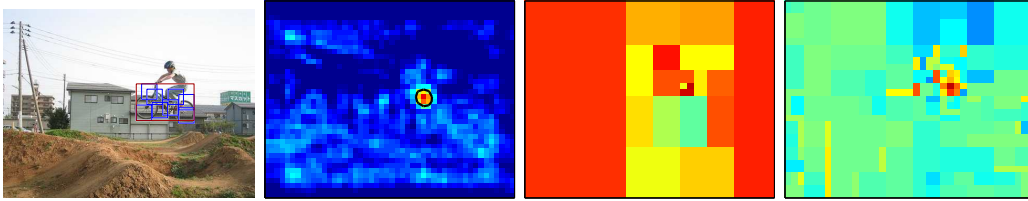

(a) Input & Detection result   (b) Detector score $S(x)$   (c) BB for $\arg\max_x S(x)$ (d) BB for $S(x) \geq -1$.

Figure 1: Motivation for Branch-and-Bound (BB) approach: standard part-based models evaluate a classifier's score $S(x)$ over the whole image domain. Typically only a tiny portion of the image domain should be positive-in (b) we draw a black contour around $\{x : S(x) > -1\}$ for an SVM-based classifier. BB ignores large intervals with low $S(x)$ by upper bounding their values, and postponing their 'exploration' in favor of more promising ones. In (c) we show as heat maps the upper bounds of the intervals visited by BB until the strongest location was explored, and in (d) of the intervals visited until all locations $x$ with $S(x) > -1$ were explored.

speeds up detection by upper bounding the score of the detector within *intervals* of $x$ while using low-cost operations. This allows us to use a prioritized search strategy that can refine these bounds on promising intervals, while postponing the exploration of less promising intervals.

This is demonstrated in Fig. 1(c,d) where we show as heat maps the upper bounds of the intervals visited by BB: parts of the image where the heat maps are more fine grained correspond to image locations that seemed promising. If our goal is to maximize $S(x)$ BB discards a huge amount of computation, as shown in (c); even with a more conservative criterion, i.e. finding all $x : S(x) > -1$ (d), a large part of the image domain is effectively ignored and the algorithm obtains refined bounds only around 'interesting' image locations.

## 3.2   Dual Trees: Data Structures for Set-Set interactions

The main technical challenge is to efficiently compute upper bounds for a model involving deformable parts; our main contribution consists in realizing that this can be accomplished with the Dual Tree data structure of [7]. We now give a high-level description of Dual Trees, leaving concrete aspects for their adaptation to the detection problem; we assume the reader is familiar with KD-trees.

Dual Trees were developed to efficiently evaluate expressions of the form:

$$P(x_j) = \sum_{i=1}^{N} w_i K(x_j, x_i), \quad x_i \in X_S, \quad i = 1, \ldots N, \quad x_j \in X_D \quad j = 1, \ldots, M \quad (2)$$

where $K(\cdot, \cdot)$ is a separable, decreasing kernel, e.g. a Gaussian with diagonal covariance. We refer to $X_S$ as 'source' terms, and to $X_D$ as 'domain' terms, the idea being that the source points $X_S$ generate a 'field' $P$, which we want evaluate at the domain locations $X_P$.

Naively performing the computation in Eq. 2 considers all source-domain interactions and takes $NM$ operations. The Dual Tree algorithm efficiently computes this sum by using two KD-trees, one ($\mathcal{S}$) for the source locations $X_S$ and another ($\mathcal{D}$) for the domain locations $X_D$. This allows for substantial speedups when computing Eq. 2 *for all domain points*, as illustrated in Fig. 2: if a 'chunk' of source points cannot affect a 'chunk' of domain points, we skip computing their domain-source point interactions.

## 4   DPM opitimization using Dual Tree Branch and Bound

Brand and Bound (BB) is a maximization algorithm for non-parametric, non-convex or even non-differentiable functions. BB searches for the interval containing the function's maximum using a prioritized search strategy; the priority of an interval is determined by the function's upper bound within it. Starting from an interval containing the whole function domain, BB increasingly narrows down to the solution: at each step an interval of solutions is popped from a priority queue, split into sub-intervals (Branch), and a new upper bound for those intervals is computed (Bound). These intervals are then inserted in the priority queue and the process repeats until a singleton interval is popped. If the bound is tight for singletons, the first singleton will be the function's global maximum.

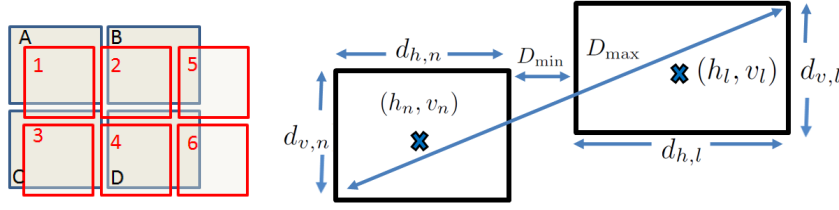

Figure 2: Left: Dual Trees efficiently deal with the interaction of 'source' (red) and 'domain' points (blue), using easily computable bounds. For instance points lying in square 6 cannot have a large effect on points in square A, therefore we do not need to go to a finer level of resolution to exactly estimate their interactions. Right: illustration of the terms involved in the geometric bound computations of Eq. 10.

Coming to our case, the DPM criterion developed in Sec. 3.1 is a sum of scores of the form:

$$s_p(x_0) = \max_{x_P} m_p(x_p, x_0) = \max_{(h_p, v_p)} U_p(h_p, v_p) - (h_p - h_0 - \eta_p)^2 H_p - (v_p - v_0 - \nu_p)^2 V_p. \quad (3)$$

Using Dual Tree terminology the 'source points' correspond to part locations $x_p$, i.e. $X_{S_p} = \{x_p\}$, and the 'domain points' to object locations $x_0$, i.e. $X_D = \{x_0\}$. Dual Trees allow us to efficiently derive bounds for $s_p(x_0), x_0 \in X_D$, the scores that a set of object locations can have due to a set of part $p$ locations. Once these are formed, we add over parts to bound the score $S(x_0) = \sum_p s_p(x_0), x_0 \in X_D$. This provides the bound needed by Branch-and Bound (BB).

We now present our approach through a series intermediate problems. These may be amenable to simpler solutions, but the more complex solutions discussed finally lead to our algorithm.

### 4.1  Maximization for One Domain Point

We first introduce notation: we index the source/domain points in $X_S/X_D$ using $i/j$ respectively. We denote by $w_i^p = U_p(x_i)$ the unary potential of part $p$ at location $x_i$. We shift the unary scores by the nominal offsets $\mu$, which gives new source locations: $x_i \rightarrow x_i - \mu_p, (h_i, v_i) \rightarrow (h_i - \eta_p, v_i - \nu_p)$. Finally, we drop $p$ from $m_p$, $H_p$ and $V_p$ unless necessary. We can now write Eq. 3 as:

$$m(h_0, v_0) = \max_{i \in S_p} w_i - H(h_i - h_0)^2 - V(v_i - v_0)^2. \quad (4)$$

To evaluate Eq. 4 at $(h_0, v_0)$ we use prioritized search over intervals of $i \in S_p$, starting from $S_p$ and gradually narrowing down to the best $i$. To prioritize intervals we use a KD-tree for the source points $x_i \in X_{S_p}$ to quickly compute bounds of Eq. 4. In specific, if $S_n$ is the set of children of the $n$-th node of the KD-tree for $S_p$, consider the subproblem:

$$m_n(h_0, v_0) = \max_{i \in S_n} w_i - H(h_i - h_0)^2 - V(v_i - v_0)^2 = \max_{i \in S_n} w_i + \mathcal{G}_i, \quad (5)$$

where $\mathcal{G}_i \doteq -H(h_i - h_0)^2 - V(v_i - v_0)^2$ stands for the geometric part of Eq. 5. We know that for all points $(h_i, v_i)$ within $S_n$ we have $h_i \in [l_n, r_n]$ and $v_i \in [b_n, t_n]$, where $l, r, b, t$ are the left, right, bottom, top axes defining $n$'s bounding box, $B_n$. We can then bound $\mathcal{G}_i$ within $S_n$ as follows:

$$\overline{\mathcal{G}_n} = -H \min(\lceil l - h_0 \rceil, \lceil h_0 - r \rceil)^2 - V \min(\lceil b - v_0 \rceil, \lceil v_0 - t \rceil)^2 \quad (6)$$

$$\underline{\mathcal{G}_n} = -H \max(l - h_0, h_0 - r)^2 - V \max(b - v_0, v_0 - t)^2, \quad (7)$$

where $\lceil \cdot \rceil = \max(\cdot, 0)$, and $\overline{\mathcal{G}_n} \geq \mathcal{G}_i \geq \underline{\mathcal{G}_n} \forall i \in S_n$. The upper bound is zero inside $B_n$ and uses the boundaries of $B_n$ that lie closest to $(h_0, v_0)$, when $(h_0, v_0)$ is outside $B_n$. The lower bound uses the distance from $(h_0, v_0)$ to the furthest point within $B_n$.

Regarding the $w_i$ term in Eq. 5, for both bounds we can use the value $w_j, j = \arg\max_{i \in S_n} w_i$. This is clearly suited for the upper bound. For the lower bound, since $\mathcal{G}_i > \underline{\mathcal{G}_n} \forall i \in S_n$, we have $\max_{i \in S_n} w_i + \mathcal{G}_i \geq w_j + \mathcal{G}_j \geq w_j + \underline{\mathcal{G}_n}$. So $w_j + \underline{\mathcal{G}_n}$ provides a proper lower bound for $\max_{i \in S_n} w_i + \mathcal{G}_i$. Summing up, we bound Eq. 5 as: $w_j + \overline{\mathcal{G}_n} \geq m_n(h_0, v_0) \geq w_j + \underline{\mathcal{G}_n}$.

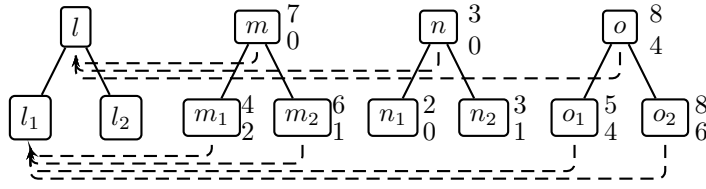

Figure 3: Supporter pruning: source nodes $\{m, n, o\}$ are among the possible supporters of domain-node $l$. Their upper and lower bounds (shown as numbers to the right of each node) are used to prune them. Here, the upper bound for $n$ (3) is smaller than the maximal lower bound among supporters (4, from $o$): this implies the upper bound of $n$'s children contributions to $l$'s children (shown here for $l_1$) will not surpass the lower bound of $o$'s children. We can thus safely remove $n$ from the supporters.

We can use the upper bound in a prioritized search for the maximum of $m(h_0, v_0)$, as described in Table 1. Starting with the root of the KD-tree we expand its children nodes, estimate their priorities-upper bounds, and insert them in a priority queue. The search stops when the first leaf node is popped; this provides the maximizer, as its upper and lower bounds coincide and all other elements waiting in queue have smaller upper bounds. The lower bound is useful in Sec. 4.2.

## 4.2 Maximization for All Domain Points

Having described how KD-trees to provide bounds in the single domain point case, we now describe how Dual Trees can speedup this operation in when treating multiple domain points simultaneously. In specific, we consider the following maximization problem:

$$x^* = \arg \max_{x \in X_D} m(x) = \arg \max_{j \in D} \max_{i \in S} w_i - H(h_i - h_j)^2 - V(v_i - v_j)^2, \tag{8}$$

where $X_D/D$ is the set of domain points/indices and $S$ are the source indices. The previous algorithm could deliver $x^*$ by computing $m(x)$ repeatedly for each $x \in X_D$ and picking the maximizer. But this will repeat similar checks for neighboring domain points, which can instead be done jointly.

For this, as in the original Dual Tree work, we build a second KD-tree for the domain points ('Domain tree', as opposed to 'Source tree'). The nodes in the Domain tree ('domain-nodes') correspond to intervals of domain points that are processed jointly. This saves repetitions of similar bounding operations, and quickly discards large domain areas with poor bounds.

For the bounding operations, as in Sec. 4.1 we consider the effect of source points contained in a node $S_n$ of the Source tree. The difference is that now we bound the maximum of this quantity over domain points contained in a domain-node $D_l$. In specific, we consider the quantity:

$$m_{l,n} = \max_{j \in D_l} \max_{i \in S_n} \quad w_i - H(h_i - h_j)^2 - V(v_i - v_j)^2 \tag{9}$$

Bounding $\mathcal{G}_{i,j} = -H(h_i - h_j)^2 - V(v_i - v_j)^2$ involves two 2D intervals, one for the domain-node $l$ and one for the domain-node $n$. If the interval for node $n$ is centered at $h_n, v_n$, and has dimensions $d_{h,n}, d_{v,n}$, we use $\bar{d}_h = \frac{1}{2}(d_{h,l} + d_{h,n})$, $\bar{d}_v = \frac{1}{2}(d_{v,l} + d_{v,n})$ and write:

$$\overline{\mathcal{G}_{l,n}} = -H \max(\lceil h_n - h_l - \bar{d}_h \rceil, \lceil h_l - h_n - \bar{d}_h \rceil)^2 - V \max(\lceil v_n - v_l - \bar{d}_v \rceil, \lceil v_l - v_n - \bar{d}_v \rceil)^2$$

$$\underline{\mathcal{G}_{l,n}} = -H \max( h_n - h_l + \bar{d}_h , h_l - h_n + \bar{d}_h )^2 - V \max( v_n - v_l - \bar{d}_v , v_l - v_n - \bar{d}_v )^2$$

We illustrate these bounds in Fig. 2. The upper bound is zero if the boxes overlap, or else equals the (scaled) distance of their closest points. The lower bound uses the furthest points of the two boxes.

As in Sec. 4.1, we use $w_n^* = \max_{i \in S_n} w_i$ for the first term in Eq. 9, and bound $m_{l,n}$ as follows:

$$\underline{\mathcal{G}_{l,n}} + w_n^* \leq m_{l,n} \leq \overline{\mathcal{G}_{l,n}} + w_n^*. \tag{10}$$

This expression bounds the maximal value $m(x)$ that a point $x$ in domain-node $l$ can have using contributions from points in source-node $n$. Our initial goal was to find the maximum using all possible source point contributions. We now describe a recursive approach to limit the set of source-nodes considered, in a manner inspired from the 'multi-recursion' approach of [7].

For this, we associate every domain-node $l$ with a set $\mathcal{S}_l$ of 'supporter' source-nodes that can yield the maximal contribution to points in $l$. We start by associating the root node of the Domain tree with the root node of the Source-tree, which means that all domain-source point interactions are originally considered.

We then recursively increase the 'resolution' of the Domain-tree in parallel with the 'resolution' of the Source-tree. More specifically, to determine the supporters for a child $m$ of domain-node $l$ we consider only the children of the source-nodes in $\mathcal{S}_l$; formally, denoting by pa and ch the parent and child operations respectively we have $\mathcal{S}_m \subset \cup_{n \in \mathcal{S}_{\mathrm{pa}(m)}}\{\mathrm{ch}(n)\}$.

Our goal is to reduce computation by keeping $\mathcal{S}_m$ small. This is achieved by pruning based on both the lower and upper bounds derived above. The main observation is that when we go from parents to children we decrease the number of source/domain points; this tightens the bounds, i.e. makes the upper bounds less optimistic and the lower bounds more optimistic. Denoting the maximal lower bound for contributions to parent node $l$ by $\underline{\mathcal{G}_l} = \max_{n \in \mathcal{S}_l} \underline{\mathcal{G}_{l,n}}$, this means that $\underline{\mathcal{G}_k} \geq \underline{\mathcal{G}_l}$ if $\mathrm{pa}(k) = l$. On the flip side, $\overline{\mathcal{G}_{l,n}} \leq \overline{\mathcal{G}_{k,q}}$ if $\mathrm{pa}(k) = l, \mathrm{pa}(q) = n$. This means that if for source-node $n$ at the parent level $\overline{\mathcal{G}_{l,n}} < \underline{\mathcal{G}_l}$, at the children level the children of $n$ will contribute something worse than $\underline{\mathcal{G}_m}$, the lower bound on $l$'s child score. We therefore do not need to keep $n$ among $\mathcal{S}_l$ - its children's contribution will be certainly worse than the best contribution from other node's children. Based on this observation we can reduce the set of supporters, while guaranteeing optimality.

Pseudocode summarizing this algorithm is provided in Table 1. The bounds in Eq. 10 are used in a prioritized search algorithm for the maximum of $m(x)$ over $x$. The algorithm uses a priority queue for Domain tree nodes, initialized with the root of the Domain tree (i.e. the whole range of possible locations $x$). At each iteration we pop a Domain tree node from the queue, compute upper bounds and supporters for its children, which are then pushed in the priority queue. The first leaf node that is popped contains the best domain location: its upper bound equals its lower bound, and all other nodes in the priority queue have smaller upper bounds, therefore cannot result in a better solution.

### 4.3 Maximization over All Domain Points and Multiple Parts: Branch and Bound for DPMs

The algorithm we described in the previous subsection is essentially a Branch-and-Bound (BB) algorithm for the maximization of a merit function

$$x^* = \arg\max_{x_0} m(x_0) = \arg \max_{(h_0, v_0)} \max_{i \in S_p} \quad w_i - H(h_i - h_0)^2 - V(v_i - v_0)^2 \qquad (11)$$

corresponding to a DPM with a single-part ($p$). To see this, recall that at each step BB pops a domain of the function being maximized from the priority queue, splits it into subdomains (Branch), and computes a new upper bound for the subdomains (Bound). In our case Branching amounts to considering the two descendants of the domain node being popped, while Bounding amounts to taking the maximum of the upper bounds of the domain node supporters.

The single-part DPM optimization problem is rather trivial, but adapting the technique to the multi-part case is now easy. For this, we rewrite Eq. 1 in a convenient form as:

$$m(h_0, v_0) = \sum_{p=0}^{P} \max_{i \in S} w_{p,i} - H_p(h_i^p - h_0)^2 - V_p(v_i^p - v_0)^2 \qquad (12)$$

using the conventions we used in Eq. 4. Namely, we only consider using points in $S$ for object parts, and subtract $m_p$ from $h_i, v_i$ to yield simple quadratic forms; since $m_p$ is part-dependent, we now have a $p$ superscript for $h_i, v_i$. Further, we have in general different $H, V$ variables for different parts, so we brought back the $p$ subscript for these. Finally, $w_{p,i}$ depends on $p$, since the same image point will give different unary potentials for different object parts.

From this form we realize that computing the upper bound of $m(x)$ within a range of values of $x$, as required by Branch-and-Bound is as easy as it was for the single terms in the previous section. In specific we have $m(x) = \sum_{p=1}^{P} m_p(x)$, where $m_p$ are the individual part contributions; since $\max_x \sum_{p=0}^{P} m_p(x) \leq \sum_{p=0}^{P} \max_x m_p(x)$. we can separately upper bound the individual part contributions, and sum them up to get an overall upper bound.

Pseudocode describing the maximization algorithm is provided in Table 1. Note that each part has its own KDtree (SourcT[p]): we build a separate Source-tree per part using the part-specific coordinates

$(h^p, v^p)$ and weights $w_{p,i}$. Each part's contribution to the score is computed using the supporters it lends to the node; the total bound is obtained by summing the individual part bounds.

---

**Single Domain Point**

```
IN: ST, x {Source Tree, Location x}
OUT: arg max_{x_i∈ST} m(x, x_i)
Push(S,ST.root);
while 1 do
   Pop(S,popped);
   if popped.UB = popped.LB then
      return popped;
   end if
   for side = [Left,Right] do
      child = popped.side;
      child.UB = BoundU(x,child);
      child.LB = BoundL(x,child);
      Push(S,child);
   end for
end while
```

**Multiple Domain Points**

```
IN: ST, DT {Source/Domain Tree}
OUT: arg max_{x∈DT} max_{i∈ST} m(x, x_i)
Seed = DT.root;
Seed.supporters = ST.Root;
Push(S,Seed);
while 1 do
   Pop(S,popped);
   if popped.UB = popped.LB then
      return popped;
   end if
   for side = [Left,Right] do
      child = popped.side;
      supp = Descend(popped.supp);
      UB,supc = Bound(child,supp,DT,ST);
      child.UB = UB;
      child.supc = supc;
      Push(S,child);
   end for
end while
```

**Multiple Domain Points, Multiple Parts**

```
IN: ST[P], DT {P Source Trees/Domain Tree}
OUT: arg max_{x∈DT} Σ_p max_{i∈ST[P]} m(x, x^p, i)
Seed = DT.root;
for p = 1 to P do
   Seed.supporters[p] = ST[p].Root;
end for
Push(S,Seed);
while 1 do
   Pop(S,popped);
   if popped.UB = popped.LB then
      return popped;
   end if
   for side = [Left,Right] do
      child = popped.side;
      UB = 0;
      for part = 1:P do
         supp = Descend(popped.supp[part])
         UP,s = Bound(child,supp,DT,ST[p]);
         child.supp[part] = s;
         UB = UB + UP;
      end for
      child.UB = UB;
      Push(S,child);
   end for
end while
```

**Bounding Routine**

```
IN: child,supporters,DT,ST
OUT: supch, LB {Chosen supporters, Max LB}
UB = -∞; LB = ∞;
for n ∈ supporters do
   UB[n] = BoundU(DT.node[child],ST.node[n]);
   LB[n] = BoundL(DT.node[child],ST.node[n]);
end for
MaxLB = max(LB);
supch = supporters(find(UB>MaxLB));
Return supch, MaxLB;
```

Table 1: Pseudocode for the algorithms presented in Section 4.

## 5    Results - Application to Deformable Object Detection

To estimate the merit of BB we first compare with the mixtures-of-DPMs developed and distributed by [3]. We directly extend the Branch-and-Bound technique that we developed for a single DPM to deal with multiple scales and mixtures ('ORs') of DPMs [4, 21], by inserting all object hypotheses into the same queue. To detect multiple instances of objects at multiple scales we continue BB after getting the best scoring object hypothesis. As termination criterion we choose to stop when we pop an interval whose upper bound is below a fixed threshold.

Our technique delivers essentially the same results as [4]. One minuscule difference is that BB uses floating point arithmetic for the part locations, while in GDT they are necessarily processed at integer resolution; other than that the results are identical. We therefore do not provide any detection performance curves, but only timing results.

Coming to time efficiency, in Fig. 4 (a) we compare the results of the original DPM mixture model and our implementation. We use 2000 images from the Pascal dataset and a mix of models for different object clases (gains vary per category). We consider the standard detection scenario where we want to detect all objects in an image having score above a certain threshold. We show how

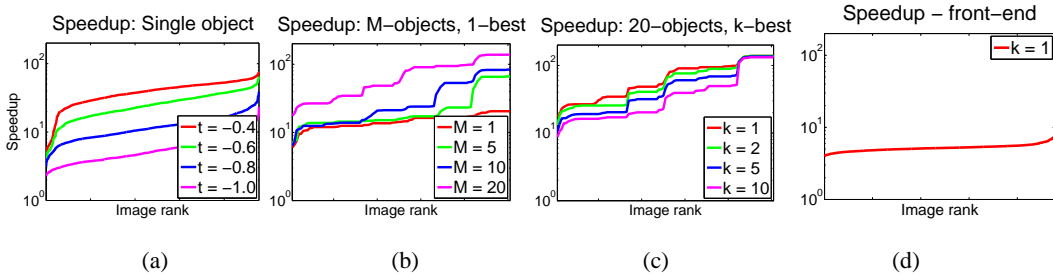

Figure 4: (a) Single-object speedup of Branch and Bound compared to GDTs on images from the Pascal dataset, (b,c) Multi-object speedup. (d) Speedup due to the front-end computation of the unary potentials. Please see text for details.

the threshold affects the speedup we obtain; for a conservative threshold the speedup is typically tenfold, but as we become more aggressive it doubles.

As a second application, we consider the problem of identifying the 'dominant' object present in the image, i.e. the category the gives the largest score. Typically simpler models, like bag-of-words classifiers are applied to this problem, based on the understanding that part-based models can be time-consuming, therefore applying a large set of models to an image would be impractical.

Our claim is that Branch-and-Bound allows us to pursue a different approach, where in fact having more object categories can *increase* the speed of detection, if we leave the unary potential computation aside. In specific, our approach can be directly extended to the multiple-object detection setting; as long as the scores computed by different object categories are commensurate, they can all be inserted in the same priority queue. In our experiments we observed that we can get a response faster by introducing more models. The reason for this is that including into our object repertoire a model giving a large score helps BB stop; otherwise BB keeps searching for another object.

In plots (b),(c) Fig. 4 we show systematic results on the Pascal dataset. We compare the time that would be required by GDT to perform detection of all multiple objects considered in Pascal, to that of a model simultaneously exploring all models. In (b) we show how finding the first-best result is accelerated as the number of objects (M) increases; while in (c) we show how increasing the 'k' in 'k-best' affects the speedup. For small values of $k$ the gains become more pronounced. Of course if we use a fixed threshold the speedup would not change, when compared to plot (a), since essentially the objects do not 'interact' in any way (we do not use nonmaximum suppression). But as we turn to the best-first problem, the speedup becomes dramatic, ranging in the order of up to a hundred times.

We note that the timings refer to the 'message passing' part implemented with GDT and not the computation of unary potentials, which is common for both models, and is currently the bottleneck. Even though it is tangential to our contribution in this paper, we mention that as shown in plot (d) we compute unary potentials approximately five times faster than the single-threaded convolution provided by [3] by exploiting Matlab's optimized matrix multiplication routines.

# 6   Conclusions

In this work we have introduced Dual-Tree Branch-and-Bound for efficient part-based detection. We have used Dual Trees to compute upper bounds on the cost function of a part-based model and thereby derived a Branch-and-Bound algorithm for detection. Our algorithm is exact and makes no approximations, delivering identical results with the DPMs used in [4], but in typically 10-15 less time. Further, we have shown that the flexibility of prioritized search allows us to consider new tasks, such as multiple-object detection, which yielded further speedups. The main challenge for future work will be to reduce the unary term computation cost; we intend to use BB for this task too.

# 7   Acknowledgements

We are grateful to the authors of [3, 12, 9] for making their code available, and to the reviewers for constructive feedback. This work was funded by grant ANR-10-JCJC -0205.

# References

[1] Y. Chen, L. Zhu, C. Lin, A. L. Yuille, and H. Zhang. Rapid inference on a novel and/or graph for object detection, segmentation and parsing. In *NIPS*, 2007.

[2] P. Felzenszwalb, D. McAllester, and D. Ramanan. A discriminatively trained, multiscale, deformable part model. In *CVPR*, 2008.

[3] P. F. Felzenszwalb, R. B. Girshick, and D. McAllester. Discriminatively trained deformable part models, release 4. http://www.cs.brown.edu/ pff/latent-release4/.

[4] P. F. Felzenszwalb, R. B. Girshick, and D. A. McAllester. Cascade object detection with deformable part models. In *CVPR*, 2010.

[5] P. F. Felzenszwalb and D. P. Huttenlocher. Distance transforms of sampled functions. Technical report, Cornell CS, 2004.

[6] V. Ferrari, M. J. Marin-Jimenez, and A. Zisserman. Progressive search space reduction for human pose estimation. In *CVPR*, 2008.

[7] A. G. Gray and A. W. Moore. Nonparametric density estimation: Toward computational tractability. In *SIAM International Conference on Data Mining*, 2003.

[8] E. Grimson. *Object Recognition by Computer*. MIT Press, 1991.

[9] A. T. Ihler, E. B. Sudderth, W. T. Freeman, and A. S. Willsky. Efficient multiscale sampling from products of gaussian mixtures. In *NIPS*, 2003.

[10] I. Kokkinos and A. Yuille. HOP: Hierarchical Object Parsing. In *CVPR*, 2009.

[11] I. Kokkinos and A. L. Yuille. Inference and learning with hierarchical shape models. *International Journal of Computer Vision*, 93(2):201–225, 2011.

[12] C. Lampert, M. Blaschko, and T. Hofmann. Beyond sliding windows: Object localization by efficient subwindow search. In *CVPR*, 2008.

[13] C. H. Lampert. An efficient divide-and-conquer cascade for nonlinear object detection. In *CVPR*, 2010.

[14] D. Lee, A. G. Gray, and A. W. Moore. Dual-tree fast gauss transforms. In *NIPS*, 2005.

[15] A. Lehmann, B. Leibe, and L. V. Gool. Fast PRISM: Branch and Bound Hough Transform for Object Class Detection. *International Journal of Computer Vision*, 94(2):175–197, 2011.

[16] V. Lempitsky, A. Blake, and C. Rother. Image segmentation by branch-and-mincut. In *ECCV*, 2008.

[17] P. Moreels, M. Maire, and P. Perona. Recognition by probabilistic hypothesis construction. In *ECCV*, page 55, 2004.

[18] M. Pedersoli, A. Vedaldi, and J. Gonzàlez. A coarse-to-fine approach for fast deformable object detection. In *CVPR*, 2011.

[19] B. Sapp, A. Toshev, and B. Taskar. Cascaded models for articulated pose estimation. In *ECCV*, 2010.

[20] P. Viola and M. Jones. Rapid Object Detection using a Boosted Cascade of Simple Features. In *CVPR*, 2001.

[21] S. C. Zhu and D. Mumford. Quest for a Stochastic Grammar of Images. *Foundations and Trends in Computer Graphics and Vision*, 2(4):259–362, 2007.

